# Neurally Inspired Plasticity in Oculomotor Processes

**Paul A. Viola**
Artificial Intelligence Laboratory
Massachusetts Institute of Technology
Cambridge, MA 02139

## ABSTRACT

We have constructed a two axis camera positioning system which is roughly analogous to a single human eye. This Artificial-Eye (A-eye) combines the signals generated by two rate gyroscopes with motion information extracted from visual analysis to stabilize its camera. This stabilization process is similar to the vestibulo-ocular response (VOR); like the VOR, A-eye learns a system model that can be incrementally modified to adapt to changes in its structure, performance and environment. A-eye is an example of a robust sensory system that performs computations that can be of significant use to the designers of mobile robots.

## 1  Introduction

We have constructed an "artificial eye" (A-eye), an autonomous robot that incorporates a two axis camera positioning system (figure 1). Like a the human oculomotor system, A-eye can estimate the rotation rate of its body with a gyroscope and estimate the rotation rate of its "eye" by measuring image slip across its "retina". Using the gyroscope to sense rotation, A-eye attempts to stabilize its camera by driving the camera motors to counteract body motion. The conversion of gyro output to motor command is dependent on the characteristics of the gyroscope, the structure of camera lensing system and the response of the motors. A correctly functioning stabilization system must model the characteristics of each of these external variables.

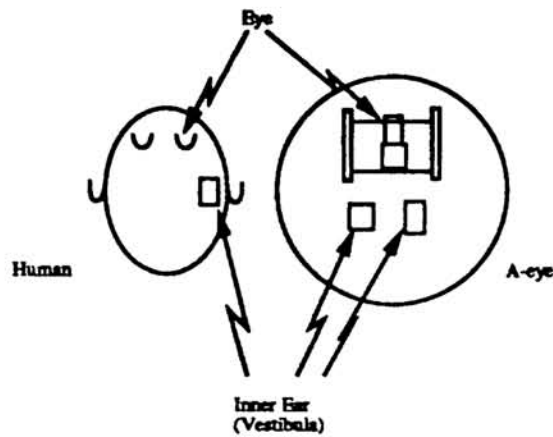

**Figure 1:** The construction of A-eye can be viewed in rough analogy to the human oculomotor system. In place of an eye, A-eye has a camera on a two axis positioning platform. In place of the circular canals of the inner ear, A-eye has two rate gyroscopes that measure rotation in perpendicular axes.

Since camera motion implies stabilization error, A-eye uses a visual estimate of camera motion to incrementally update its system model. When the camera is correctly stabilized there is no statistically significant slip. Whenever a particular gyro measurement is associated with a result camera motion, A-eye makes an incremental change to its response to that particular measurement to reduce that error in the future.

A-eye was built for two reasons: to facilitate the operation of complex visually guided mobile robots and to explore the applicability of simple learning techniques to the construction of a robust robot.

## 2 Autonomous Robots

An autonomous robot must function correctly for long periods of time without human intervention. It is certainly difficult to create an autonomous robot or process that will function accurately, both initially and perpetually. To achieve such a goal, autonomous processes must be able to adapt both to unforeseen aspects of the environment and inaccuracies in construction. One approach to attaining successful autonomous performance would entail the full characterization of the robot's structure, its performance requirements, and its relationship with the environment. Since clearly both the robot and its environment are susceptible to change any characterization could not be static. In contrast, our approach only partially categorizes the robot's structure, environment, and task. Without more detailed information initial performance is inaccurate. However, by using a measure of error in performance initially partial categorization can incrementally improved. In addition, a change to system performance can be compensated continually. In this way the extensive analysis and engineering that would be required to characterize, foresee

and circumvent variability can be greatly reduced.

## 3   The VOR

The oculomotor processes found in vertebrates are well studied examples of adaptive, visually guided processing [Gou85]. The three oculomotor processes found almost universally in vertebrates (the vestibulo-ocular response, the optokinetic system, and the saccadic system), accurately perform ocular positioning tasks with little or no conscious direction. The response times of these systems demonstrate that little high level, "conscious", processing could take place. In a limited sense these processes are autonomous, and it should come as no surprise that they are quite plastic. Such plasticity is necessary to counteract the foreseeable changes in the eye due to growth and aging and the unforeseeable changes due to illness and injury.

The VOR works to counteract the motion of a creature in its environment. A correctly functioning VOR ensures that a creature "sees" as little unintended motion as possible. Miles [FAM81] and others have demonstrated that the VOR is an adaptive motor response, capable of significant recalibration in a matter of days. Adaptation can be demonstrated by the use of inverting or magnifying spectacles. While wearing these glasses the correct orbital motion of the eye, given a particular head motion, is significantly different from the normal response. Initially, the response to head motion is an incorrect eye motion. With time eye motion begins to approach the correct counteracting motion. This kind of adaptation allows an animal to continue functioning in spite of injury or illness.

## 4   The Device

A-eye is a small autonomous robot that incorporates a CCD camera, a three wheel base, a two axis pitch/yaw camera positioning platform, and two rate gyroscopes. On board processing includes a Motorola microcontroller and 68020 based video processing board. Including batteries, A-eye is a foot high cylinder that is 12 inches wide. In its present configuration A-eye can run autonomously for up to three hours (figure 2).

A-eye's goal is to learn how to keep its camera stable as its base trundles down corridors. There are two sources of information regarding the motion of A-eye's base: gyro rotation measurements and optical flow. Rate gyroscopes measure base rotation rate directly. Visual analysis can be used to estimate motion by a number of methods of varying complexity (see [Hil83] for a good overview). By attempting to measure only camera rotation from slip complexity can be avoided. The simple method we have chosen measures the slip of images across the retina.

### 4.1   Visual Rotation Estimation

Our approach to camera rotation estimation uses a pre-processing subunit commonly known as a "Reichard detector" which for clarity we will call a *shift and correlate unit* [PR73]. A *shift and correlate unit* has as its inputs a set of samples

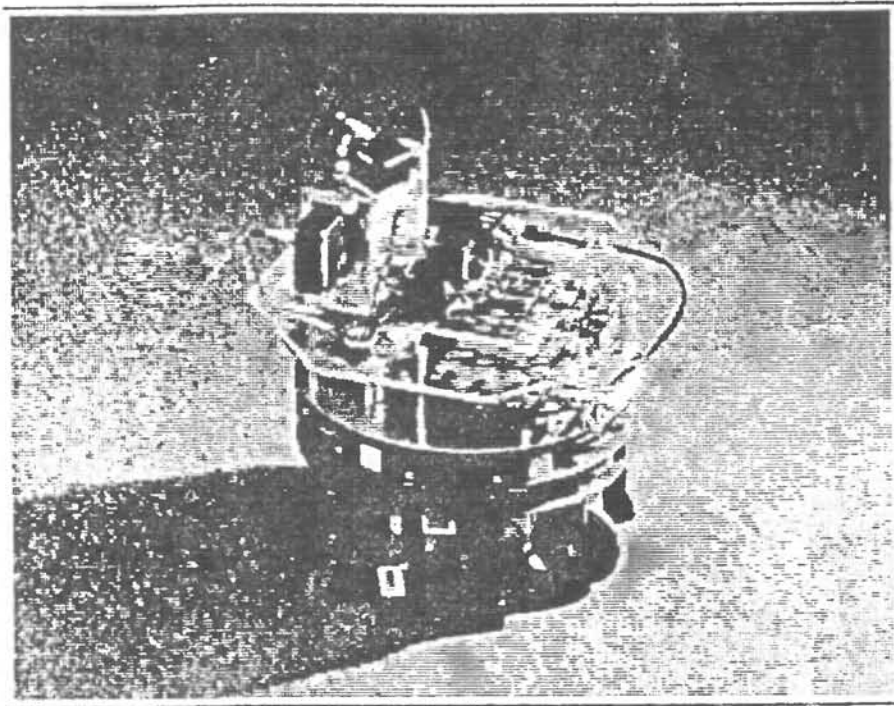

Figure 2: A photo of the current state of A-eye.

from a blurred area of the retina. It shifts these inputs spatially and correlates them with a previous, unshifted set of inputs. When two succeeding images are identical except for a spatial shift, the units which perform that shift respond strongly. Clearly the activity of a *shift and correlate unit* contains information about retinal motion. Due to the size and direction of shift, some detectors will be sensitive to small motions, others large motions, and each will be sensitive to a particular direction of motion.

The input from the *shift and correlate units* is used to build value-unit encoded retinal velocity map, in which each unit is sensitive to a different direction and range of velocities. The map has 9 units in a 3 by 3 grid (fig 3). To create such a map, each of the *shift and correlate units* is connected to every map unit. By moving the camera, displaced images that are examples of motion, are generated. The motor command that generated this motion example corresponds to a unit in the visual velocity map. Connection weights are updated by a standard least squares learning rule. In operation, the most active unit represents the estimate of visual motion.

## 4.2   Gyroscope Rotation Estimation

Contrary to first intuition, vertebrates do not rely on visual information to stabilize their eyes. Instead head rotation information measured by the inner ear, or the vestibula, is used keep the eyes stable. Animals do not measure ocular motion directly from visual information for two reasons: a) the response rates of photoreceptors prevent useful visual processing during rapid eye movements [Gou85] b) the

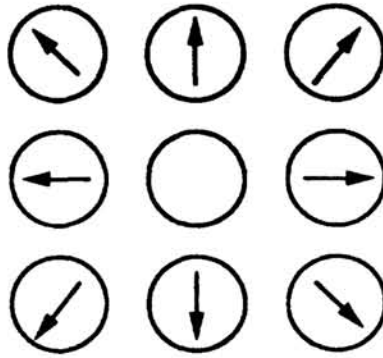

**Figure 3:** The 9 unit velocity map has 1 unit for each of the 8 "chess moves".

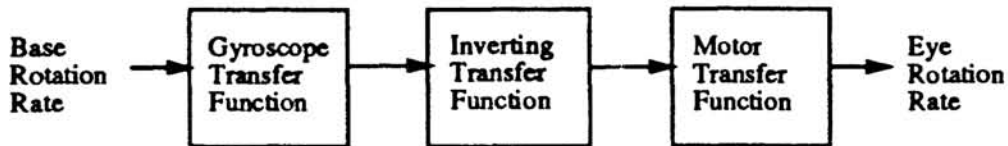

**Figure 4:** Open-loop control of ocular position based on gyroscope output.

required visual analysis takes approximately 100ms[1]. These difficulties combine to prevent rapid response to unexpected head and body motions. A-eye is beset with similar limitations and we have chosen a similar solution.

The output of the gyroscope is some function of head rotation rate. Stabilization is achieved by driving the ocular motors directly in opposition to the measured velocity (fig 4). This counteract rotation of the base in one direction by moving the camera in the opposite direction. Such an open-loop system is very simple and can perform well; they are unfortunately very reliant on proper calibration and recalibration to maintain performance [Oga70].

A-eye maintains calibration information in the form of a function from gyroscope output to motor velocity command. This function is an 8 unit gaussian radial basis approximation network [TP89]. Basis function approximation has excellent computational properties while representing wide variety of smooth functions. Weights are modified with a simple least squares update rule, based errors in camera motion detected visually.

## 5    Training A-eye

A-eye learns to perform the VOR in a two phase process. First, the measurement of visual motion is calibrated to the generation of camera motion commands. Second,

the stabilizing motor responses to gyroscope measurements are approximated. This approximation is modified based on a visual estimate of camera motion.

By observing motor commands and comparing them to the resulting visual motion, a map from visual motion to appropriate motor command can be learned. To train the visual motion map, A-eye performs a set of characteristic motions and observes the results. Each motor command is categorized as one of the 9 distinct motions encoded by the visual motion map. With each motion, the connections from *shift and correlate units* to the visual motion map are updated so that issuing a motor command results in activity in the correct visual motion unit. Because no reference is made to external variables, this measure of visual motion is completely relative to the function of the camera motors. The visual motion map plays the role of error signal for later learning.

By observing both the gyroscope output and the visual response from head motion, A-eye learns the appropriate compensating eye motion for all head motions. Eye compensation motions are the result of motor commands generated by the approximation network applied to the gyroscope output. Incorrect responses will cause visual motion. This motion, as measured by the visual motion detector, is the error signal that drives the modification of the approximation network. This is the heart of the adaptation in the VOR.

## 5.1   Results

While training the motion detector and approximation network there are 5 training events per second (the visual analysis takes about 200 msec). Training the visual motion detector can take up to 10 minutes (in a few environments the weights refuse to settle on the correct values). While it is possible to hand wire a detector that is 95% accurate, most learned detectors worked well, attaining 85% accuracy. In both cases, the detectors have the desirable capability of rejecting object motion whenever there is actual camera motion (this is due to the global nature of the analysis).

The approximation network converges to a function that performs well in minutes (figure 5). Analysis of the images generated by the camera leads us to bound the cumulative error in rotation over a 1 minute trial at 5 degrees (we believe this approaches the accuracy limitations inherent in the gyroscope).

An approach to reducing this gyroscope error involves yet another oculomotor process: optokinetic nystagmus (OKN). This is the tendency for an otherwise undirected eye to follow visual motion in the absence of vestibular cues. A-eye's visual motion map is in motor coordinates. By directing the camera in the opposite direction from observed motion, residual errors in VOR can be reduced.

## 6   Application

We claim that the stabilization that results from a correctly calibrated VOR is useful both for navigation and scene analysis. A stable inertial reference can act to assist tactical navigation when traversing rough terrain. Large body attitude

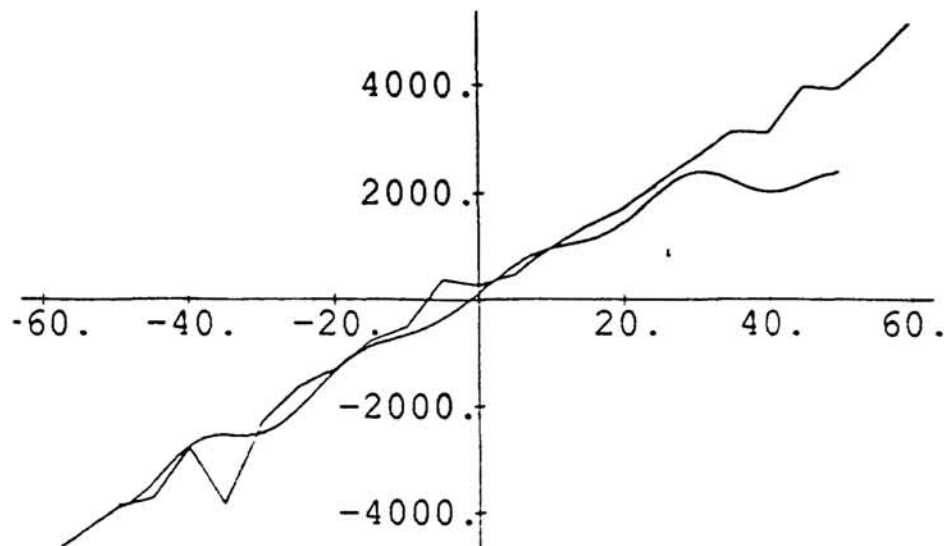

Figure 5: A correct transfer function (rough) and the learned (smoother) approximation.

changes, that can result from such travel, make it difficult to maintain a navigational bearing. However, when there exists a relatively stable inertial reference frame less analysis need be performed to predict or sense changes in bearing by other means.

The VOR is especially applicable to legged vehicles, where the terrain and the form of locomotion can cause constant rapid changes in attitude [Rai89] [Ang89]. The task of adapting conventional vision systems to such vehicles is formidable. As the rate of pitching increases, the quality of video images degrade, while the task of finding a correspondence between successive images will increase in complexity. With the addition of the visual stabilization that A-eye can provide, an otherwise complex visual analysis task can be much simplified.

## 7   Conclusions

A-eye is in part a response to the observation that static calibration is a disastrous weakness. Static calibration not only forces an engineer to expend additional effort at design time, it requires constant performance monitoring and recalibration. By creating a device that monitors its own performance and adapts to changes, significant work can be saved in design and at numerous times during the lifetime of the device.

A-eye is also in part a confirmation that simple, tractable and reliable learning mechanisms are sufficient to perform useful motor learning.

Finally, A-eye is in part a demonstration that useful visual processing can be performed in real-time with an reasonable amount of computation. This processing yields the additional side-benefit of simplifying the complex task of visual recognition.

## Acknowledgements

This report describes research done at the Artificial Intelligence Laboratory of the Massachusetts Institute of Technology. Support for this research was provided by Hughes Artificial Intelligence Center contract #SI-804475-D, the Office of Naval Research contract N00014-86-K-0685, and the Defense Advanced Research Projects Agency under Office of Naval Research contract N00014-85-K-0124.

## Footnotes

[1] Ocular following, the tendency to follow the motion of a scene in the absence of head motion, has a typical latency of 100ms [FM87].

# References

[Ang89] Colin Angle. Genghis, a six legged autonomous walking robot. Master's thesis, MIT, 1989.

[FAM81] S. G. Lisberger F. A. Miles. Plasticity in the vestibulo-ocular reflex: A new hypothesis. *Ann. Rev. Neurosci.*, 4:273–299, 1981.

[FM87] K. Kawano F.A. Miles. Visual stabilization of the eyes. *TINS*, 4(10):153–158, 1987. Reference on Opto-kinetic nystagmus latency.

[Gou85] Peter Gouras. Oculomotor system. In James Schwartz Eric Kandel, editor, *Principles of Neuroscience*, chapter 34. Elsevier Science Publishing, 1985.

[Hil83] Ellen C. Hildreth. *The Measurement of Visual Motion*. The MIT Press, 1983. Good book on the extraction of motion from edges.

[Oga70] Katsuhiko Ogata. *Modern Control Engineering*. Prentice-Hall, Englewood Cliffs, N.J., 1970. Steady State Frequency Response (page 372).

[PR73] T. Poggio and W. Reichard. Considerations on models of movement detection. *Kybernetic*, 13:223–227, 1973.

[Rai89] Marc H. Raibert. Trotting, pacing, and bounding by a quadruped robot. *Journal of Biomechanics*, 1989.

[TP89] Federico Girosi Tomaso Poggio. A theory of networks for approximation and learning. AI Memo 1140, MIT, 1989.
